# RecNorm: Simultaneous Normalisation and Classification applied to Speech Recognition

**John S. Bridle**
Royal Signals and Radar Est.
Great Malvern
UK   WR14 3PS

**Stephen J. Cox**
British Telecom Research Labs.
Ipswich
UK   IP5 7RE

## Abstract

A particular form of neural network is described, which has terminals for acoustic patterns, class labels and speaker parameters. A method of training this network to "tune in" the speaker parameters to a particular speaker is outlined, based on a trick for converting a supervised network to an unsupervised mode. We describe experiments using this approach in isolated word recognition based on whole-word hidden Markov models. The results indicate an improvement over speaker-independent performance and, for unlabelled data, a performance close to that achieved on labelled data.

## 1   INTRODUCTION

We are concerned to emulate some aspects of perception. In particular, the way that a stimulus which is ambiguous, perhaps because of unknown lighting conditions, can become unambiguous in the context of other such stimuli: the fact that they are subject to the same unknown conditions gives our perceptual apparatus enough constraints to solve the problem. Individual words are often ambiguous even to human listeners. For instance a Cockney might say the word "ace" to sound the same as a Standard English speaker's "ice". Similarly with "room" and "rum", or "work" and "walk" in other pairs of British English accents. If we heard one of these ambiguous pronunciations, knowing nothing else about the speaker we could not tell which word had been said. For current automatic speech recognition (ASR) systems such effects are much more frequent, because we do not know how to concentrate on the important aspects of the signal locally, nor how to exploit the fact that some unknown properties apply to whole words, nor how to bring to bear on the task of

acoustic disambiguation all the information that is normally latent in the context of the utterance.

Most attempts to construct ASR systems which can be used by many persons have used so-called speaker-independent models. When decoding a short sequence of words there is no way of imposing our knowledge that all the speech is uttered by one person.

To enable adaptation using small amounts of speech from a new speaker we propose to factor the speech knowledge into speaker-independent models, continous speaker-specific parameters and a transformation which modifies the models according to the speaker parameters. (In this paper we shall only use transformations which can just as easily be applied to the input patterns.) We are specially interested in the possibility of estimating such parameters from quite small amounts of *unlabelled* speech, such as a few short words or one longer word. Although the types of models and transformations we have used are very simple, we hope the general approach will be applicable to quite sophisticated models and transformations which will be necessary for future high-performance speech recognition systems.

# 2   AN ADAPTIVE NETWORK APPROACH

## 2.1   GENERAL IDEA

Suppose we had a feed-forward network with three (vector-valued) terminals, which encapsulates our knowledge of the relationship between acoustic patterns, $X$, class labels (e.g. word identities) $C$, and speaker parameters, $Q$.

Training such a network seems difficult, because although we can supply $(X,C)$ pairs, we do not know the appropriate values of $Q$. (We only know the *names* of the speakers, or perhaps some phonetician's descriptive labels.)

In training the network we start with default values of $Q$, feed forward from $X$ and $Q$ to $C$, back-propagate derivatives to internal parameters of the network (weights, transition probabilities, etc.) and also to the $Q$s, enforcing the constraint that the $Q$s for any one speaker stay equal. We can imagine one copy of the network for each utterance, with the $Q$ terminals of networks dealing with the same speaker strapped together. One convenient implementation (for a small number of training speakers) is to adapt one $Q$ vector per speaker in a set of weights from one-from-N coded speaker identity inputs to linear units, as we shall see later.

Once the network is trained we have two modes of use. If we have available one or more known utterances by a new speaker, then we can "tune-in" to the speaker (as during training) except that only the $Q$ inputs are adjusted. The case of most interest in this paper, however, is when we have a few *unknown* words from an unknown speaker. We set up a $Q$-strapped set of networks, one for each word, initialise the $Q$ values to their defaults, propagate forwards to produce a set of distributions across word labels, and then we use a technique which tends to sharpen these distributions. In the simplest case, the sharpening process could be a matter of: for each utterance pick the word label with the largest output, and assuming it to be correct back-propagate derivatives to the common $Q$. In practice, we can use a gentler method in which large outputs get most 'encouragement'. For some

networks it is possible to show that such a "phantom target" procedure can lead to hillclimbing on the likelihood of the data given an assumption about the form of the generator of the data (see Appendix).

## 2.2   SIMPLE NETWORK ILLUSTRATION

We have explored these ideas using a very simple network based on that in figure 1. It can be viewed either as a feedforward network with radial (minus Euclidean distance squared) units and a generalised-logistic (Softmax) output non-linearity, or as a Gaussian classifier in which the covariance matrices are unit diagonal (see [Bri90b]). Training is done by gradient-based optimisation, using back-propagation of partial derivatives. During training the criterion is based on relative entropy (likelihood of the targets given the network outputs) [Bri90c]. (Such *discriminative* training can lead to different results from the usual model-based methods [Bri90b], which in this case would set the reference points at the data means for each class.)

This simple classifier network is preceded by a full linear transformation (6 parameters), so the equivalent model-based classifier has Gaussian distributions with the same arbitrary covariance matrix for each class. We use the biasses of the linear units as speaker parameters, so the weights from speaker identity inputs go straight into the hidden units, is as figure 2.

During adaptation to a new speaker from *unlabelled* tokens, the speaker parameters of the transformation are allowed to adapt, but the ("phantom") targets are derived from the outputs themselves (the targets are just double the outputs) so that the largest outputs are encouraged.

In figure 3 we see the adaptation of the positions of the reference points of the radial units in figure 2 when the input points are essentially the 6 reference points displaced to one side (to represent one example of each word spoken by a new speaker). Adaptation based on tentative classifications pulls the reference points towards a position where the inputs can be given confident, consistent labels.

## 3   SPEECH RECOGNITION EXPERIMENTS

We have applied these ideas to the problem of recognising a few short, confusable words from a known set, all spoken by the same unknown speaker. If our method works we should be able to recognise each word better (on average) if we also look at a few other unknown words from the same speaker.

The dataset [Sal89], which had been recorded previously for other purposes, comprised the British English isolated pronounciations of the names of the letters of the alphabet, each spoken 3 times by each speaker. The 104 speakers were divided into two groups of 52 (Train and Test), balanced for age and sex. Initial acoustic analysis produced 28-component spectrum vectors, 100 per second. In place of the 2-D input patterns discussed above, each speech pattern was a variable-duration sequence (typically 50) of 28-vectors.

In place of each simple Gaussian density class-models we used a set of Gaussian densities and a matrix of probabilities of transitions between them. Each class-model is thus a hidden Markov model (HMM) of a word. We used 26 HMMs, each

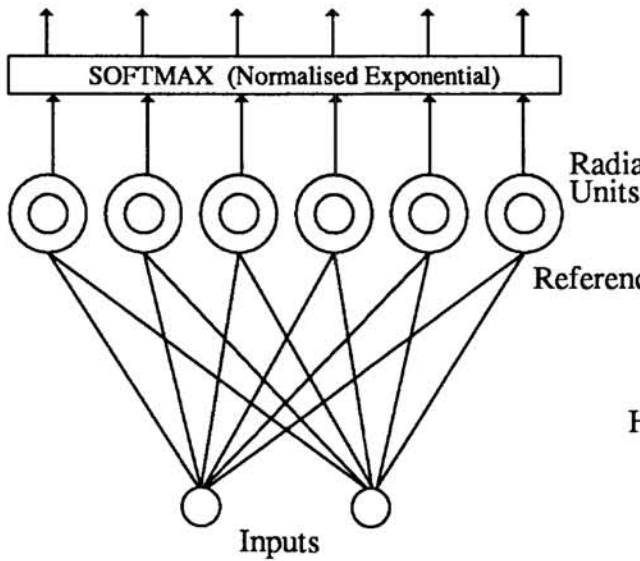

Fig.1  Feedforward Network
Implementing Simple Gaussian
Classifier

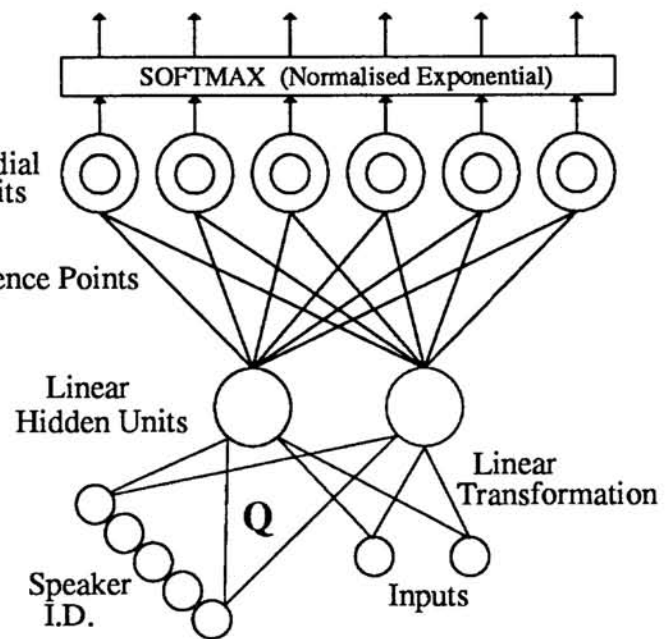

Fig.2  Gaussian classifier network with
input transformation and speaker inputs

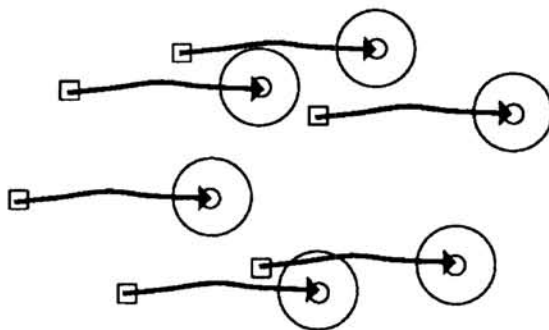

Fig.3  Adaptation to 6 displaced
points

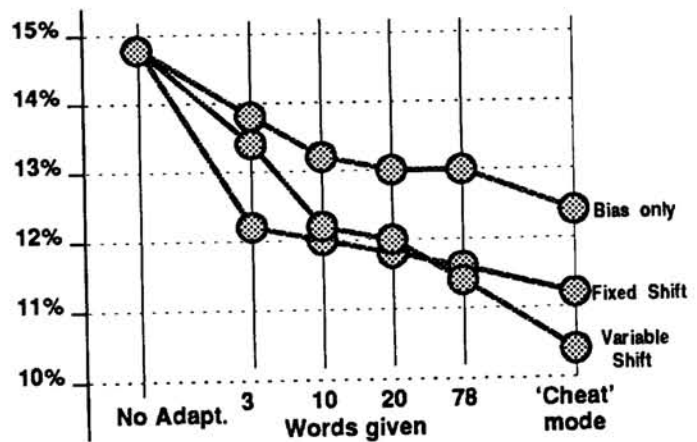

Fig.4  Average error rates for alphabet
word recognition

with 15 states, each with a 3-component Gaussian mixture output distribution. For further details see [CB90].

The equivalent to the evaluation of a Gaussian density in the simple network is the Forward (or Alpha) computation of the likelihood of the data given a (hidden Markov) model. This calculation can be thought of as being performed by a recurrent network of a special form. When we include the Bayes inversion to produce probabilities of the classes (this is a normalisation if we assume equal prior probabilities) we obtain the equivalent of the simple network of figure 1, which we call an Alphanet[Bri90a].

In place of the 2-component linear transformation in figure 2 we use a constrained linear transformation based on [Hun81] $y_i = a_i x_{i-1} + b_i x_i + c_i x_{i+1} + d_i$, where $x_i, i = 1, \ldots 28$, is the log spectrum amplitude in frequency channel $i$.

We tried three conditions:

- Bias Only: $a = 0$, $b = 1$, $c = 0$ (28 parameters)

- Fixed Shift: $a_i = a$, $b_i = b$, $c_i = c$ (31 parameters)

- Variable Shift: the general case (107 parameters)

Figure 4 shows average word error rates for the three types of transformation, for different numbers of utterances taken together ($N = 3, 10, 20, 78$). $N = 1$ is the non-adaptive case. 'Cheat' Mode is a check on the power of the transformations: for each test speaker, all 78 utterances were used to set the parameters of the transformation, then recognition performance was measured using those parameters of the same utterances.

We see:

- Use of unsupervised adaptation reduced the error rates.

- The reductions are not spectacular (15% errors to 12% errors, a reduction in error rate by 20%.) but they are statistically significant and may be practically significant too.

- The performance in 'Cheat' Mode is only a little better than in unsupervised mode, so performance is being limited by the power of the transformation.

- The Fixed Shift transformation gives quite good results even on only 3 words at a time.

When tested on a 120 talker telephone-line database of isolated digits collected at British Telecom, the best unsupervised speaker adaptation technique gave a 37% decrease in error-rate (for both supervised and unsupervised adaptation on 5 utterances) using a simple front-end consisting of 8 MFCCs (mel frequency-scale cepstrum coefficients). A more sophisticated front-end (using differential information and energy) improved the unadapted performance by 63% over the 8 MFCC front-end. Using this front-end, the best unsupervised adaptation technique (on 5 utterances) decreased the error-rate by a further 25%

# 4  CONCLUSIONS

The results reported here show that simultaneous word recognition and speaker normalisation can be made to work, that it improves performance over the corresponding speaker-independent version, and that given 3 to 10 unknown words performance can be almost as good as when the adaptation is done using knowledge of the word identities. The main extensions we are interested in are to use non-linear transformations, and to learn low-dimensional but effective speaker parameterisations.

# A  Unsupervised Adaptation using Phantom Targets

We aim to motivate the 'phantom target' trick of feeding back twice each output of the network as a target.

Suppose we have a classifier network, with a 1-from-N output coding, and a Softmax output nonlinearity. We write $Q_j$ for an output value, $V_j$ for an input to the Softmax output stage, $x$ for the input to the network, $c$ for a class and $\theta$ for parameters which we may want to adjust. A typical output value is

$$Q_j(x, \theta) = e^{V_j(x, \theta)} \Big/ \sum_k e^{V_k(x, \theta)}.$$

The output values are interpretable as estimates of posterior probabilities: $Q_j \approx \Pr(c = j \mid x, \theta)$. For the next step we assume there are some implicit probability density functions $P_j(x, \theta) \approx \Pr(x \mid c = j, \theta)$ Assuming equal prior probabilities of the classes for simplicity, Bayes rule gives

$$Q_j(x, \theta) = P_j(x, \theta) \Big/ \sum_{k=1}^{N} P_k(x, \theta),$$

so we suppose that

$$P_j(x, \theta) = \frac{1}{z_j(\theta)} e^{V_j(x, \theta)},$$

where the normalisation is

$$z_j(\theta) = \int e^{V_j(x, \theta)} dx.$$

In the networks we use, the same normalisation applies to all the classes, so we write $z_j(\theta) = z(\theta)$.

A maximum-likelihood approach to unsupervised adaptation maximises the likelihood of the data given the set of (equally probable) distributions, which is

$$P(x, \theta) = \sum_{k=1}^{N} P_k(x, \theta) \frac{1}{N},$$

It is simpler to maximise the *log* likelihood:

$$L(x, \theta) \stackrel{\triangle}{=} \log P(x, \theta) = \log \sum_k P_k(x, \theta) - \log N = \log \sum_k e^{V_k(x, \theta)} - \log z(\theta) - \log N.$$

We shall need

$$\frac{\partial L}{\partial V_j} = \frac{1}{\sum_k e^{V_k(\boldsymbol{x}, \boldsymbol{\theta})}} e^{V_j(\boldsymbol{x}, \boldsymbol{\theta})} - \frac{1}{z(\boldsymbol{\theta})} \frac{\partial z(\boldsymbol{\theta})}{\partial V_j(\boldsymbol{x}, \boldsymbol{\theta})}.$$

(The likelihood of the whole training set is the product of the likelihoods of the individual patterns, and the log turns the product into a sum, so we can sum the derivatives of $L$ over the training set.)

We can often assume that the normalisation is independent of $\boldsymbol{\theta}$, giving

$$\frac{\partial L}{\partial V_j} = \frac{e^{V_j(\boldsymbol{x}, \boldsymbol{\theta})}}{\sum_k e^{V_k(\boldsymbol{x}, \boldsymbol{\theta})}} = Q_j(\boldsymbol{x}, \boldsymbol{\theta}).$$

If we have a supervised backprop network using the relative entropy based criterion (rather than squared error) [?], we are minimising $J = -\sum_j T_j \log Q_j$, where $T_j$ is the target for the $j$th output. We know [Bri90b] that $\frac{\partial J}{\partial V_j} = Q_j - T_j$, so if we set $T_j = 2Q_j$ we have $\frac{\partial J}{\partial V_,} = -\frac{\partial L}{\partial V_,}$, and minimising $J$ is equivalent to maximising $L$.

For the simple Gaussian network of figure 1, this unsupervised adaptation, applied to the reference points, can be understood as an on-line, gradient descent, relative of the k-means cluster analysis procedure, or of the LBG vector quantiser design method, or indeed of Kohonen's feature map (without the neighbourhood constraints).

# References

[Bri90a] J S Bridle. Alphanets: a recurrent 'neural' network architecture with a hidden Markov model interpretation. *Speech Communication*, Special "Neurospeech" issue, February 1990.

[Bri90b] J S Bridle. Probabilistic interpretation of feedforward classification network outputs, with relationships to statistical pattern recognition. In F Fougelman-Soulie and J Hérault, editors, *Neuro-computing: algorithms, architectures and applications*, NATO ASI Series on Systems and computer science. Springer-Verlag, 1990.

[Bri90c] J S Bridle. Training stochastic model recognition algorithms as networks can lead to maximum mutual information estimation of parameters. In *Advances in Neural Information Processing Systems 2*. Morgan Kaufmann, 1990.

[CB90] S J Cox and J S Bridle. Simultaneous speaker normalisation and utterance labelling using Bayesian/neural net techniques. In *Proc. IEEE Int. Conf. Acoustics Speech and Signal Processing*, 1990.

[Hun81] M J Hunt. Speaker adaptation for word-based speech recognition. *J. Acoust. Soc. Amer*, 69:S41–S42, 1981. (abstract only).

[Sal89] J A S Salter. The RT5233 Alphabetic database for the Connex project. Technical Report RT52/G231/89, BT Technology Executive, 1989.